# Learning a Continuous Hidden Variable Model for Binary Data

**Daniel D. Lee**
Bell Laboratories
Lucent Technologies
Murray Hill, NJ 07974
ddlee@bell-labs.com

**Haim Sompolinsky**
Racah Institute of Physics and
Center for Neural Computation
Hebrew University
Jerusalem, 91904, Israel
haim@fiz.huji.ac.il

## Abstract

A directed generative model for binary data using a small number of hidden continuous units is investigated. A clipping nonlinearity distinguishes the model from conventional principal components analysis. The relationships between the correlations of the underlying continuous Gaussian variables and the binary output variables are utilized to learn the appropriate weights of the network. The advantages of this approach are illustrated on a translationally invariant binary distribution and on handwritten digit images.

## Introduction

Principal Components Analysis (PCA) is a widely used statistical technique for representing data with a large number of variables [1]. It is based upon the assumption that although the data is embedded in a high dimensional vector space, most of the variability in the data is captured by a much lower dimensional manifold. In particular for PCA, this manifold is described by a linear hyperplane whose characteristic directions are given by the eigenvectors of the correlation matrix with the largest eigenvalues. The success of PCA and closely related techniques such as Factor Analysis (FA) and PCA mixtures clearly indicate that much real world data exhibit the low dimensional manifold structure assumed by these models [2, 3].

However, the linear manifold structure of PCA is not appropriate for data with binary valued variables. Binary values commonly occur in data such as computer bit streams, black-and-white images, on-off outputs of feature detectors, and electrophysiological spike train data [4]. The Boltzmann machine is a neural network model that incorporates hidden binary spin variables, and in principle, it should be able to model binary data with arbitrary spin correlations [5]. Unfortunately, the

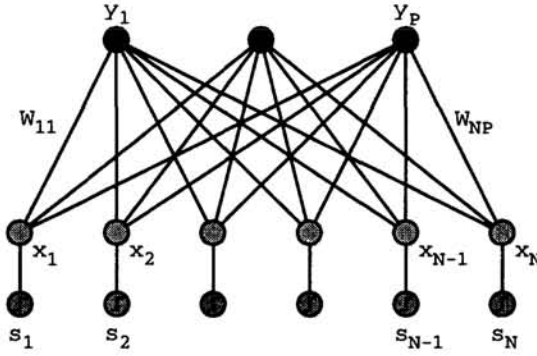

Figure 1: Generative model for $N$-dimensional binary data using a small number $P$ of continuous hidden variables.

computational time needed for training a Boltzmann machine renders it impractical for most applications.

In these proceedings, we present a model that uses a small number of continuous hidden variables rather than hidden binary variables to capture the variability of binary valued visible data. The generative model differs from conventional PCA because it incorporates a clipping nonlinearity. The resulting spin configurations have an entropy related to the number of hidden variables used, and the resulting states are connected by small numbers of spin flips. The learning algorithm is particularly simple, and is related to PCA by a scalar transformation of the correlation matrix.

## Generative Model

Figure 1 shows a schematic diagram of the generative process. As in PCA, the model assumes that the data is generated by a small number $P$ of continuous hidden variables $y_i$. Each of the hidden variables are assumed to be drawn independently from a normal distribution with unit variance:

$$P(y_i) = \exp(-y_i^2/2)/\sqrt{2\pi}. \tag{1}$$

The continuous hidden variables are combined using the feedforward weights $W_{ij}$, and the $N$ binary output units are then calculated using the sign of the feedforward activations:

$$x_i = \sum_{j=1}^{P} W_{ij} y_j \tag{2}$$

$$s_i = \text{sgn}(x_i). \tag{3}$$

Since binary data is commonly obtained by thresholding, it seems reasonable that a proper generative model should incorporate such a clipping nonlinearity. The generative process is similar to that of a sigmoidal belief network with continuous hidden units at zero temperature. The nonlinearity will alter the relationship between the correlations of the binary variables and the weight matrix $W$ as described below.

The real-valued Gaussian variables $x_i$ are exactly analogous to the visible variables of conventional PCA. They lie on a linear hyperplane determined by the span of the matrix $W$, and their correlation matrix is given by:

$$C^{xx} = \langle xx^T \rangle = WW^T. \tag{4}$$

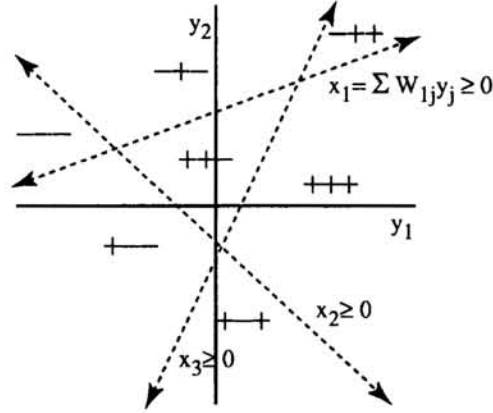

Figure 2: Binary spin configurations $s_i$ in the vector space of continuous hidden variables $y_j$ with $P = 2$ and $N = 3$.

By construction, the correlation matrix $C^{xx}$ has rank $P$ which is much smaller than the number of components $N$. Now consider the binary output variables $s_i = \text{sgn}(x_i)$. Their correlations can be calculated from the probability distribution of the Gaussian variables $x_i$:

$$(C^{ss})_{ij} = \langle s_i s_j \rangle = \int \prod_k dy_k \, P(x_k) \, \text{sgn}(x_i) \, \text{sgn}(x_j) \qquad (5)$$

where

$$P(\mathbf{x}) = \frac{1}{(2\pi)^{N/2}\sqrt{|C^{xx}|}} \exp\left(-\frac{1}{2}\mathbf{x}^T (C^{xx})^{-1}\mathbf{x}\right). \qquad (6)$$

The integrals in Equation 5 can be done analytically, and yield the surprisingly simple result:

$$(C^{ss})_{ij} = \left(\frac{2}{\pi}\right) \sin^{-1}\left[\frac{C^{xx}_{ij}}{\sqrt{C^{xx}_{ii} C^{xx}_{jj}}}\right]. \qquad (7)$$

Thus, the correlations of the clipped binary variables $C^{ss}$ are related to the correlations of the corresponding Gaussian variables $C^{xx}$ through the nonlinear arcsine function. The normalization in the denominator of the arcsine argument reflects the fact that the sign function is unchanged by a scale change in the Gaussian variables.

Although the correlation matrix $C^{ss}$ and the generating correlation matrix $C^{xx}$ are easily related through Equation 7, they have qualitatively very different properties. In general, the correlation matrix $C^{ss}$ will no longer have the low rank structure of $C^{xx}$. As illustrated by the translationally invariant example in the next section, the spectrum of $C^{ss}$ may contain a whole continuum of eigenvalues even though $C^{xx}$ has only a few nonzero eigenvalues.

PCA is typically used for dimensionality reduction of real variables; can this model be used for compressing the binary outputs $s_i$? Although the output correlations $C^{ss}$ no longer display the low rank structure of the generating $C^{xx}$, a more appropriate measure of data compression is the entropy of the binary output states. Consider how many of the $2^N$ possible binary states will be generated by the clipping process. The equation $x_i = \sum_j W_{ij}y_j = 0$ defines a $P-1$ dimensional hyperplane in the $P$-dimensional state space of hidden variables $y_j$, which are shown as dashed lines in Figure 2. These hyperplanes partition the half-space where $s_i = +1$ from the

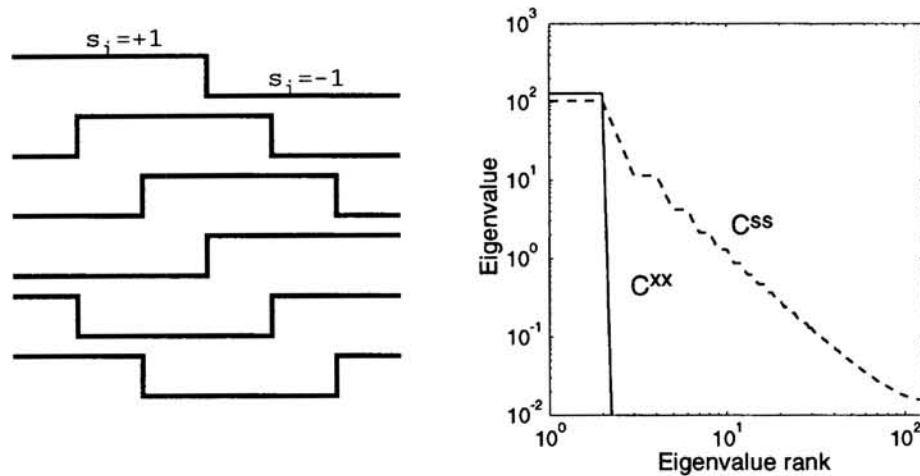

Figure 3: Translationally invariant binary spin distribution with $N = 256$ units. Representative samples from the distribution are illustrated on the left, while the eigenvalue spectrum of $C^{ss}$ and $C^{xx}$ are plotted on the right.

region where $s_i = -1$. Each of the $N$ spin variables will have such a dividing hyperplane in this $P$-dimensional state space, and all of these hyperplanes will generically be unique. Thus, the total number of spin configurations $s_i$ is determined by the number of cells bounded by $N$ dividing hyperplanes in $P$ dimensions. The number of such cells is approximately $N^P$ for $N \gg P$, a well-known result from perceptrons [6]. To leading order for large $N$, the entropy of the binary states generated by this process is then given by $S = P \log N$. Thus, the entropy of the spin configurations generated by this model is directly proportional to the number of hidden variables $P$.

How is the topology of the binary spin configurations $s_i$ related to the PCA manifold structure of the continuous variables $x_i$? Each of the generated spin states is represented by a polytope cell in the $P$ dimensional vector space of hidden variables. Each polytope has at least $P + 1$ neighboring polytopes which are related to it by a single or small number of spin flips. Therefore, although the state space of binary spin configurations is discrete, the continuous manifold structure of the underlying Gaussian variables in this model is manifested as binary output configurations with low entropy that are connected with small Hamming distances.

## Translationally Invariant Example

In principle, the weights $W$ could be learned by applying maximum likelihood to this generative model; however, the resulting learning algorithm involves analytically intractable multi-dimensional integrals. Alternatively, approximations based upon mean field theory or importance sampling could be used to learn the appropriate parameters [7]. However, Equation 7 suggests a simple learning rule that is also approximate, but is much more computationally efficient [8]. First, the binary correlation matrix $C^{ss}$ is computed from the data. Then the empirical $C^{ss}$ is mapped into the appropriate Gaussian correlation matrix using the nonlinear transformation: $C^{xx} = \sin(\pi C^{ss}/2)$. This results in a Gaussian correlation matrix where the variances of the individual $x_i$ are fixed at unity. The weights $W$ are then calculated using the conventional PCA algorithm. The correlation matrix $C^{xx}$ is diagonalized, and the eigenvectors with the largest eigenvalues are used to form the columns of

$W$ to yield the best low rank approximation $C^{xx} \approx WW^T$. Scaling the variables $x_i$ will result in a correlation matrix $C^{xx}$ with slightly different eigenvalues but with the same rank.

The utility of this transformation is illustrated by the following simple example. Consider the distribution of $N = 256$ binary spins shown in Figure 3. Half of the spins are chosen to be positive, and the location of the positive bump is arbitrary under the periodic boundary conditions. Since the distribution is translationally invariant, the correlations $C_{ij}^{ss}$ depend only on the relative distance between spins $|i - j|$. The eigenvectors are the Fourier modes, and their eigenvalues correspond to their overlap with a triangle wave. The eigenvalue spectrum of $C^{ss}$ is plotted in Figure 3 as sorted by their rank. In this particular case, the correlation matrix $C^{ss}$ has $N/2$ positive eigenvalues with a corresponding range of values.

Now consider the matrix $C^{xx} = \sin(\pi C^{ss}/2)$. The eigenvalues of $C^{xx}$ are also shown in Figure 3. In contrast to the many different eigenvalues $C^{ss}$, the spectrum of the Gaussian correlation matrix $C^{xx}$ has only two positive eigenvalues, with all the rest exactly equal to zero. The corresponding eigenvectors are a cosine and sine function. The generative process can thus be understood as a linear combination of the two eigenmodes to yield a sine function with arbitary phase. This function is then clipped to yield the positive bump seen in the original binary distribution.

In comparison with the eigenvalues of $C^{ss}$, the eigenvalue spectrum of $C^{xx}$ makes obvious the low rank structure of the generative process. In this case, the original binary distribution can be constructed using only $P = 2$ hidden variables, whereas it is not clear from the eigenvalues of $C^{ss}$ what the appropriate number of modes is. This illustrates the utility of determining the principal components from the calculated Gaussian correlation matrix $C^{xx}$ rather than working directly with the observable binary correlation matrix $C^{ss}$.

## Handwritten Digits Example

This model was also applied to a more complex data set. A large set of $16 \times 16$ black and white images of handwritten twos were taken from the US Post Office digit database [9]. The pixel means and pixel correlations were directly computed from the images. The generative model needs to be slightly modified to account for the non-zero means in the binary outputs. This is accomplished by adding fixed biases $\xi_i$ to the Gaussian variables $x_i$ before clipping:

$$s_i = \text{sgn}(\xi_i + x_i). \tag{8}$$

The biases $\xi_i$ can be related to the means of the binary outputs through the expression:

$$\xi_i = \sqrt{2C_{ii}^{xx}} \, \text{erf}^{-1}\langle s_i \rangle. \tag{9}$$

This allows the biases to be directly computed from the observed means of the binary variables. Unfortunately, with non-zero biases, the relationship between the Gaussian correlations $C^{xx}$ and binary correlations $C^{ss}$ is no longer the simple expression found in Equation 7. Instead, the correlations are related by the following integral equation:

$$C_{ij}^{ss} = \langle s_i \rangle \langle s_j \rangle + \frac{2}{\pi} \int_0^{\frac{C_{ij}^{xx}}{\sqrt{C_{ii}^{xx} C_{jj}^{xx}}}} d\rho \, \frac{1}{\sqrt{1 - \rho^2}} \exp\left[ -\frac{1}{2(1 - \rho^2)}(\xi_i^2 + \xi_j^2 - 2\rho\xi_i\xi_j) \right]. \tag{10}$$

Given the empirical pixel correlations $C^{ss}$ for the handwritten digits, the integral in Equation 10 is numerically solved for each pair of indices to yield the appropriate

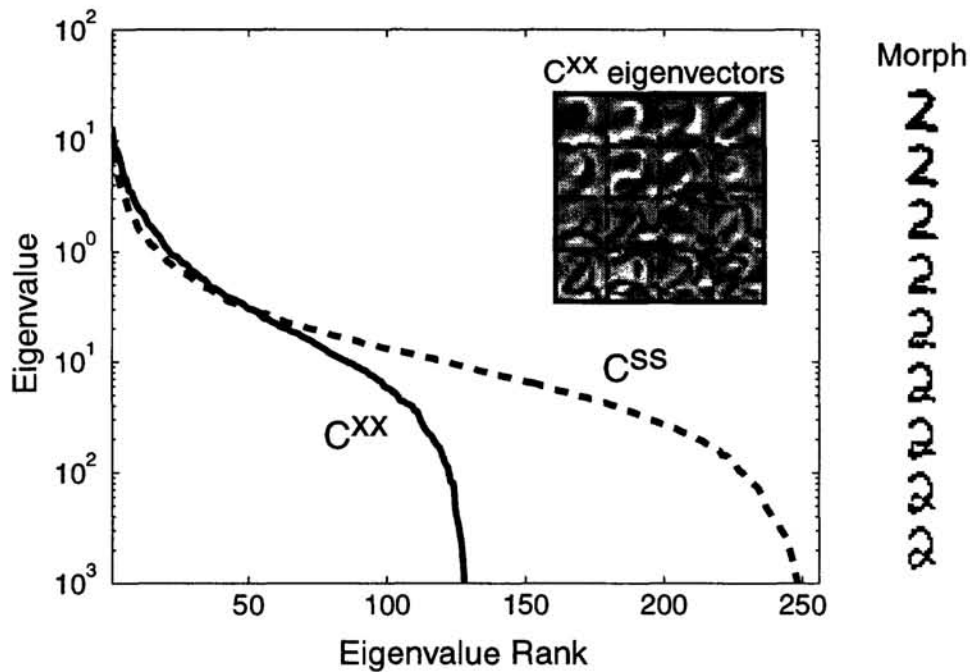

Figure 4: Eigenvalue spectrum of $C^{ss}$ and $C^{xx}$ for handwritten images of twos. The inset shows the $P = 16$ most significant eigenvectors for $C^{xx}$ arranged by rows. The right side of the figure shows a nonlinear morph between two different instances of a handwritten two using these eigenvectors.

Gaussian correlation matrix $C^{xx}$. The correlation matrices are diagonalized and the resulting eigenvalue spectra are shown in Figure 4. The eigenvalues for $C^{xx}$ again exhibit a characteristic drop that is steeper than the falloff in the spectrum of the binary correlations $C^{ss}$. The corresponding eigenvectors of $C^{xx}$ with the 16 largest positive eigenvalues are depicted in the inset of Figure 4. These eigenmodes represent common image distortions such as rotations and stretching and appear qualitatively similar to those found by the standard PCA algorithm.

A generative model with weights $W$ corresponding to the $P = 16$ eigenvectors shown in Figure 4 is used to fit the handwritten twos, and the utility of this nonlinear generative model is illustrated in the right side of Figure 4. The top and bottom images in the figure are two different examples of a handwritten two from the data set, and the generative model is used to morph between the two examples. The hidden values $y_i$ for the original images are first determined for the different examples, and the intermediate images in the morph are constructed by linearly interpolating in the vector space of the hidden units. Because of the clipping nonlinearity, this induces a nonlinear mapping in the outputs with binary units being flipped in a particular order as determined by the generative model. In contrast, morphing using conventional PCA would result in a simple linear interpolation between the two images, and the intermediate images would not look anything like the original binary distribution [10].

The correlation matrix $C^{xx}$ also happens to contain some small negative eigenvalues. Even though the binary correlation matrix $C^{ss}$ is positive definite, the transformation in Equation 10 does not guarantee that the resulting matrix $C^{xx}$ will also be positive definite. The presence of these negative eigenvalues indicates a shortcoming of the generative processs for modelling this data. In particular, the clipped Gaussian model is unable to capture correlations induced by global

constraints in the data. As a simple illustration of this shortcoming in the generative model, consider the binary distribution defined by the probability density: $P(\{s\}) \propto \lim_{\beta \to \infty} \exp(-\beta \sum_{ij} s_i s_j)$. The states in this distribution are defined by the constraint that the sum of the binary variables is exactly zero: $\sum_i s_i = 0$. Now, for $N \geq 4$, it can be shown that it is impossible to find a Gaussian distribution whose visible binary variables match the negative correlations induced by this sum constraint.

These examples illustrate the value of using the clipped generative model to learn the correlation matrix of the underlying Gaussian variables rather than using the correlations of the outputs directly. The clipping nonlinearity is convenient because the relationship between the hidden variables and the output variables is particularly easy to understand. The learning algorithm differs from other nonlinear PCA models and autoencoders because the inverse mapping function need not be explicitly learned [11, 12]. Instead, the correlation matrix is directly transformed from the observable variables to the underlying Gaussian variables. The correlation matrix is then diagonalized to determine the appropriate feedforward weights. This results in a extremely efficient training procedure that is directly analogous to PCA for continuous variables.

We acknowledge the support of Bell Laboratories, Lucent Technologies, and the US-Israel Binational Science Foundation. We also thank H. S. Seung for helpful discussions.

# References

[1] Jolliffe, IT (1986). *Principal Component Analysis*. New York: Springer-Verlag.

[2] Bartholomew, DJ (1987). *Latent variable models and factor analysis*. London: Charles Griffin & Co. Ltd.

[3] Hinton, GE, Dayan, P & Revow, M (1996). Modeling the manifolds of images of handwritten digits. *IEEE Transactions on Neural networks* **8**, 65–74.

[4] Van Vreeswijk, C, Sompolinsky, H, & Abeles, M. (1999). Nonlinear statistics of spike trains. In preparation.

[5] Ackley, DH, Hinton, GE, & Sejnowski, TJ (1985). A learning algorithm for Boltzmann machines. *Cognitive Science* **9**, 147–169.

[6] Cover, TM (1965). Geometrical and statistical properties of systems of linear inequalities with applications in pattern recognition. *IEEE Trans. Electronic Comput.* **14**, 326–334.

[7] Tipping, ME (1999). Probabilistic visualisation of high-dimensional binary data. *Advances in Neural Information Processing Systems* **11**.

[8] Christoffersson, A (1975). Factor analysis of dichotomized variables. *Psychometrika* **40**, 5–32.

[9] LeCun, Y *et al.* (1989). Backpropagation applied to handwritten zip code recognition. *Neural Computation* **1**, 541–551.

[10] Bregler, C, & Omohundro, SM (1995). Nonlinear image interpolation using manifold learning. *Advances in Neural Information Processing Systems* **7**, 973–980.

[11] Hastie, T and Stuetzle, W (1989). Principal curves. *Journal of the American Statistical Association* **84**, 502–516.

[12] Demers, D, & Cottrell, G (1993). Nonlinear dimensionality reduction. *Advances in Neural Information Processing Systems* **5**, 580–587.
